# Direct Importance Estimation with Model Selection and Its Application to Covariate Shift Adaptation

**Masashi Sugiyama**
Tokyo Institute of Technology
sugi@cs.titech.ac.jp

**Shinichi Nakajima**
Nikon Corporation
nakajima.s@nikon.co.jp

**Hisashi Kashima**
IBM Research
hkashima@jp.ibm.com

**Paul von Bünau**
Technical University Berlin
buenau@cs.tu-berlin.de

**Motoaki Kawanabe**
Fraunhofer FIRST
nabe@first.fhg.de

## Abstract

A situation where training and test samples follow different input distributions is called *covariate shift*. Under covariate shift, standard learning methods such as maximum likelihood estimation are no longer consistent—weighted variants according to the ratio of test and training input densities are consistent. Therefore, accurately estimating the density ratio, called the *importance*, is one of the key issues in covariate shift adaptation. A naive approach to this task is to first estimate training and test input densities separately and then estimate the importance by taking the ratio of the estimated densities. However, this naive approach tends to perform poorly since density estimation is a hard task particularly in high dimensional cases. In this paper, we propose a direct importance estimation method that does not involve density estimation. Our method is equipped with a natural cross validation procedure and hence tuning parameters such as the kernel width can be objectively optimized. Simulations illustrate the usefulness of our approach.

## 1 Introduction

A common assumption in supervised learning is that training and test samples follow the *same* distribution. However, this basic assumption is often violated in practice and then standard machine learning methods do not work as desired. A situation where the input distribution $P(\boldsymbol{x})$ is different in the training and test phases but the conditional distribution of output values, $P(y|\boldsymbol{x})$, remains unchanged is called *covariate shift* [8]. In many real-world applications such as robot control [10], bioinformatics [1], spam filtering [3], brain-computer interfacing [9], or econometrics [5], covariate shift is conceivable and thus learning under covariate shift is gathering a lot of attention these days.

The influence of covariate shift could be alleviated by weighting the log likelihood terms according to the *importance* [8]: $w(\boldsymbol{x}) = p_{\text{te}}(\boldsymbol{x})/p_{\text{tr}}(\boldsymbol{x})$, where $p_{\text{te}}(\boldsymbol{x})$ and $p_{\text{tr}}(\boldsymbol{x})$ are test and training input densities. Since the importance is usually unknown, the key issue of covariate shift adaptation is how to accurately estimate the importance.

A naive approach to importance estimation would be to first estimate the training and test densities separately from training and test input samples, and then estimate the importance by taking the ratio of the estimated densities. However, density estimation is known to be a hard problem particularly in high-dimensional cases. Therefore, this naive approach may not be effective—directly estimating the importance *without* estimating the densities would be more promising.

Following this spirit, the kernel mean matching (KMM) method has been proposed recently [6], which directly gives importance estimates without going through density estimation. KMM is shown

to work well, given that tuning parameters such as the kernel width are chosen appropriately. Intuitively, model selection of importance estimation algorithms (such as KMM) is straightforward by cross validation (CV) over the performance of subsequent learning algorithms. However, this is highly unreliable since the ordinary CV score is heavily biased under covariate shift—for unbiased estimation of the prediction performance of subsequent learning algorithms, the CV procedure itself needs to be importance-weighted [9]. Since the importance weight has to have been fixed when model selection is carried out by importance weighted CV, it can not be used for model selection of importance estimation algorithms.

The above fact implies that model selection of importance estimation algorithms should be performed *within* the importance estimation step in an unsupervised manner. However, since KMM can only estimate the values of the importance at training input points, it can not be directly applied in the CV framework; an out-of-sample extension is needed, but this seems to be an open research issue currently.

In this paper, we propose a new importance estimation method which can overcome the above problems, i.e., the proposed method directly estimates the importance without density estimation *and* is equipped with a natural model selection procedure. Our basic idea is to find an importance estimate $\widehat{w}(\boldsymbol{x})$ such that the Kullback-Leibler divergence from the true test input density $p_{\text{te}}(\boldsymbol{x})$ to its estimate $\widehat{p}_{\text{te}}(\boldsymbol{x}) = \widehat{w}(\boldsymbol{x})p_{\text{tr}}(\boldsymbol{x})$ is minimized. We propose an algorithm that can carry out this minimization without explicitly modeling $p_{\text{tr}}(\boldsymbol{x})$ and $p_{\text{te}}(\boldsymbol{x})$. We call the proposed method the *Kullback-Leibler Importance Estimation Procedure* (KLIEP). The optimization problem involved in KLIEP is convex, so the unique global solution can be obtained. Furthermore, the solution tends to be sparse, which contributes to reducing the computational cost in the test phase.

Since KLIEP is based on the minimization of the Kullback-Leibler divergence, its model selection can be naturally carried out through a variant of likelihood CV, which is a standard model selection technique in density estimation. A key advantage of our CV procedure is that, not the training samples, but the *test input samples* are cross-validated. This highly contributes to improving the model selection accuracy since the number of training samples is typically limited while test input samples are abundantly available.

The simulation studies show that KLIEP tends to outperform existing approaches in importance estimation including the logistic regression based method [2], and it contributes to improving the prediction performance in covariate shift scenarios.

## 2 New Importance Estimation Method

In this section, we propose a new importance estimation method.

### 2.1 Formulation and Notation

Let $\mathcal{D} \subset (\mathbb{R}^d)$ be the input domain and suppose we are given i.i.d. training input samples $\{\boldsymbol{x}_i^{\text{tr}}\}_{i=1}^{n_{\text{tr}}}$ from a training input distribution with density $p_{\text{tr}}(\boldsymbol{x})$ and i.i.d. test input samples $\{\boldsymbol{x}_j^{\text{te}}\}_{j=1}^{n_{\text{te}}}$ from a test input distribution with density $p_{\text{te}}(\boldsymbol{x})$. We assume that $p_{\text{tr}}(\boldsymbol{x}) > 0$ for all $\boldsymbol{x} \in \mathcal{D}$. Typically, the number $n_{\text{tr}}$ of training samples is rather small, while the number $n_{\text{te}}$ of test input samples is very large. The goal of this paper is to develop a method of estimating the *importance* $w(\boldsymbol{x})$ from $\{\boldsymbol{x}_i^{\text{tr}}\}_{i=1}^{n_{\text{tr}}}$ and $\{\boldsymbol{x}_j^{\text{te}}\}_{j=1}^{n_{\text{te}}}$:

$$w(\boldsymbol{x}) = \frac{p_{\text{te}}(\boldsymbol{x})}{p_{\text{tr}}(\boldsymbol{x})}.$$

Our key restriction is that we avoid estimating densities $p_{\text{te}}(\boldsymbol{x})$ and $p_{\text{tr}}(\boldsymbol{x})$ when estimating the importance $w(\boldsymbol{x})$.

### 2.2 Kullback-Leibler Importance Estimation Procedure (KLIEP)

Let us model the importance $w(\boldsymbol{x})$ by the following linear model:

$$\widehat{w}(\boldsymbol{x}) = \sum_{\ell=1}^{b} \alpha_\ell \varphi_\ell(\boldsymbol{x}), \tag{1}$$

where $\{\alpha_\ell\}_{\ell=1}^b$ are parameters to be learned from data samples and $\{\varphi_\ell(\boldsymbol{x})\}_{\ell=1}^b$ are basis functions such that

$$\varphi_\ell(\boldsymbol{x}) \geq 0 \text{ for all } \boldsymbol{x} \in \mathcal{D} \text{ and for } \ell = 1, 2, \ldots, b.$$

Note that $b$ and $\{\varphi_\ell(\boldsymbol{x})\}_{\ell=1}^b$ could be dependent on the samples $\{\boldsymbol{x}_i^{\text{tr}}\}_{i=1}^{n_{\text{tr}}}$ and $\{\boldsymbol{x}_j^{\text{te}}\}_{j=1}^{n_{\text{te}}}$, i.e., *kernel* models are also allowed—we explain how the basis functions $\{\varphi_\ell(\boldsymbol{x})\}_{\ell=1}^b$ are chosen in Section 2.3.

Using the model $\widehat{w}(\boldsymbol{x})$, we can estimate the test input density $p_{\text{te}}(\boldsymbol{x})$ by

$$\widehat{p}_{\text{te}}(\boldsymbol{x}) = \widehat{w}(\boldsymbol{x})p_{\text{tr}}(\boldsymbol{x}).$$

We determine the parameters $\{\alpha_\ell\}_{\ell=1}^b$ in the model (1) so that the Kullback-Leibler divergence from $p_{\text{te}}(\boldsymbol{x})$ to $\widehat{p}_{\text{te}}(\boldsymbol{x})$ is minimized:

$$\begin{aligned}
\text{KL}[p_{\text{te}}(\boldsymbol{x})\|\widehat{p}_{\text{te}}(\boldsymbol{x})] &= \int_{\mathcal{D}} p_{\text{te}}(\boldsymbol{x}) \log \frac{p_{\text{te}}(\boldsymbol{x})}{\widehat{w}(\boldsymbol{x})p_{\text{tr}}(\boldsymbol{x})} d\boldsymbol{x} \\
&= \int_{\mathcal{D}} p_{\text{te}}(\boldsymbol{x}) \log \frac{p_{\text{te}}(\boldsymbol{x})}{p_{\text{tr}}(\boldsymbol{x})} d\boldsymbol{x} - \int_{\mathcal{D}} p_{\text{te}}(\boldsymbol{x}) \log \widehat{w}(\boldsymbol{x}) d\boldsymbol{x}.
\end{aligned}$$

Since the first term in the last equation is independent of $\{\alpha_\ell\}_{\ell=1}^b$, we ignore it and focus on the second term. We denote it by $J$:

$$\begin{aligned}
J &= \int_{\mathcal{D}} p_{\text{te}}(\boldsymbol{x}) \log \widehat{w}(\boldsymbol{x}) d\boldsymbol{x} \qquad (2) \\
&\approx \frac{1}{n_{\text{te}}} \sum_{j=1}^{n_{\text{te}}} \log \widehat{w}(\boldsymbol{x}_j^{\text{te}}) = \frac{1}{n_{\text{te}}} \sum_{j=1}^{n_{\text{te}}} \log \left( \sum_{\ell=1}^b \alpha_\ell \varphi_\ell(\boldsymbol{x}_j^{\text{te}}) \right),
\end{aligned}$$

where the empirical approximation based on the test input samples $\{\boldsymbol{x}_j^{\text{te}}\}_{j=1}^{n_{\text{te}}}$ is used from the first line to the second line above. This is our objective function to be maximized with respect to the parameters $\{\alpha_\ell\}_{\ell=1}^b$, which is concave [4]. Note that the above objective function only involves the test input samples $\{\boldsymbol{x}_j^{\text{te}}\}_{j=1}^{n_{\text{te}}}$, i.e., we did not use the training input samples $\{\boldsymbol{x}_i^{\text{tr}}\}_{i=1}^{n_{\text{tr}}}$ yet. As shown below, $\{\boldsymbol{x}_i^{\text{tr}}\}_{i=1}^{n_{\text{tr}}}$ will be used in the constraint.

$\widehat{w}(\boldsymbol{x})$ is an estimate of the importance $w(\boldsymbol{x})$ which is non-negative by definition. Therefore, it is natural to impose $\widehat{w}(\boldsymbol{x}) \geq 0$ for all $\boldsymbol{x} \in \mathcal{D}$, which can be achieved by restricting

$$\alpha_\ell \geq 0 \text{ for } \ell = 1, 2, \ldots, b.$$

In addition to the non-negativity, $\widehat{w}(\boldsymbol{x})$ should be properly normalized since $\widehat{p}_{\text{te}}(\boldsymbol{x})\,(=\widehat{w}(\boldsymbol{x})p_{\text{tr}}(\boldsymbol{x}))$ is a probability density function:

$$\begin{aligned}
1 = \int_{\mathcal{D}} \widehat{p}_{\text{te}}(\boldsymbol{x}) d\boldsymbol{x} &= \int_{\mathcal{D}} \widehat{w}(\boldsymbol{x}) p_{\text{tr}}(\boldsymbol{x}) d\boldsymbol{x} \qquad (3) \\
&\approx \frac{1}{n_{\text{tr}}} \sum_{i=1}^{n_{\text{tr}}} \widehat{w}(\boldsymbol{x}_i^{\text{tr}}) = \frac{1}{n_{\text{tr}}} \sum_{i=1}^{n_{\text{tr}}} \sum_{\ell=1}^b \alpha_\ell \varphi_\ell(\boldsymbol{x}_i^{\text{tr}}),
\end{aligned}$$

where the empirical approximation based on the training input samples $\{\boldsymbol{x}_i^{\text{tr}}\}_{i=1}^{n_{\text{tr}}}$ is used from the first line to the second line above.

Now our optimization criterion is summarized as follows.

$$\operatorname*{maximize}_{\{\alpha_\ell\}_{\ell=1}^b} \left[ \sum_{j=1}^{n_{\text{te}}} \log \left( \sum_{\ell=1}^b \alpha_\ell \varphi_\ell(\boldsymbol{x}_j^{\text{te}}) \right) \right]$$

$$\text{subject to } \sum_{i=1}^{n_{\text{tr}}} \sum_{\ell=1}^b \alpha_\ell \varphi_\ell(\boldsymbol{x}_i^{\text{tr}}) = n_{\text{tr}} \text{ and } \alpha_1, \alpha_2, \ldots, \alpha_b \geq 0.$$

This is a convex optimization problem and the global solution can be obtained, e.g., by simply performing gradient ascent and feasibility satisfaction iteratively. A pseudo code is described in Figure 1-(a). Note that the solution $\{\widehat{\alpha}_\ell\}_{\ell=1}^b$ tends to be *sparse* [4], which contributes to reducing the computational cost in the test phase. We refer to the above method as *Kullback-Leibler Importance Estimation Procedure* (KLIEP).

**Input:** $m = \{\varphi_\ell(\boldsymbol{x})\}_{\ell=1}^{b}$, $\{\boldsymbol{x}_i^{\mathrm{tr}}\}_{i=1}^{n_{\mathrm{tr}}}$, and $\{\boldsymbol{x}_j^{\mathrm{te}}\}_{j=1}^{n_{\mathrm{te}}}$
**Output:** $\widehat{w}(\boldsymbol{x})$

$A_{j,\ell} \longleftarrow \varphi_\ell(\boldsymbol{x}_j^{\mathrm{te}})$;
$b_\ell \longleftarrow \frac{1}{n_{\mathrm{tr}}}\sum_{i=1}^{n_{\mathrm{tr}}} \varphi_\ell(\boldsymbol{x}_i^{\mathrm{tr}})$;
Initialize $\boldsymbol{\alpha}\ (> \boldsymbol{0})$ and $\varepsilon\ (0 < \varepsilon \ll 1)$;
**Repeat until convergence**
$\quad \boldsymbol{\alpha} \longleftarrow \boldsymbol{\alpha} + \varepsilon \boldsymbol{A}^\top (\boldsymbol{1}./\boldsymbol{A}\boldsymbol{\alpha})$;
$\quad \boldsymbol{\alpha} \longleftarrow \boldsymbol{\alpha} + (1 - \boldsymbol{b}^\top \boldsymbol{\alpha})\boldsymbol{b}/(\boldsymbol{b}^\top \boldsymbol{b})$;
$\quad \boldsymbol{\alpha} \longleftarrow \max(\boldsymbol{0}, \boldsymbol{\alpha})$;
$\quad \boldsymbol{\alpha} \longleftarrow \boldsymbol{\alpha}/(\boldsymbol{b}^\top \boldsymbol{\alpha})$;
**end**
$\widehat{w}(\boldsymbol{x}) \longleftarrow \sum_{\ell=1}^{b} \alpha_\ell \varphi_\ell(\boldsymbol{x})$;

(a) KLIEP main code

**Input:** $\mathcal{M} = \{m_k \mid m_k = \{\varphi_\ell^{(k)}(\boldsymbol{x})\}_{\ell=1}^{b^{(k)}}\}$,
$\quad\quad \{\boldsymbol{x}_i^{\mathrm{tr}}\}_{i=1}^{n_{\mathrm{tr}}}$, and $\{\boldsymbol{x}_j^{\mathrm{te}}\}_{j=1}^{n_{\mathrm{te}}}$
**Output:** $\widehat{w}(\boldsymbol{x})$

Split $\{\boldsymbol{x}_j^{\mathrm{te}}\}_{j=1}^{n_{\mathrm{te}}}$ into $R$ disjoint subsets $\{\mathcal{X}_r^{\mathrm{te}}\}_{r=1}^{R}$;
**for** each model $m \in \mathcal{M}$
$\quad$ **for** each split $r = 1, \ldots, R$
$\quad\quad \widehat{w}_r(\boldsymbol{x}) \longleftarrow \mathrm{KLIEP}(m, \{\boldsymbol{x}_i^{\mathrm{tr}}\}_{i=1}^{n_{\mathrm{tr}}}, \{\mathcal{X}_j^{\mathrm{te}}\}_{j \neq r})$;
$\quad\quad \widehat{J}_r(m) \longleftarrow \frac{1}{|\mathcal{X}_r^{\mathrm{te}}|}\sum_{\boldsymbol{x} \in \mathcal{X}_r^{\mathrm{te}}} \log \widehat{w}_r(\boldsymbol{x})$;
$\quad$ **end**
$\quad \widehat{J}(m) \longleftarrow \frac{1}{R}\sum_{r=1}^{R} \widehat{J}_r(m)$;
**end**
$\widehat{m} \longleftarrow \mathrm{argmax}_{m \in \mathcal{M}} \widehat{J}(m)$;
$\widehat{w}(\boldsymbol{x}) \longleftarrow \mathrm{KLIEP}(\widehat{m}, \{\boldsymbol{x}_i^{\mathrm{tr}}\}_{i=1}^{n_{\mathrm{tr}}}, \{\boldsymbol{x}_j^{\mathrm{te}}\}_{j=1}^{n_{\mathrm{te}}})$;

(b) KLIEP with model selection

Figure 1: KLIEP algorithm in pseudo code. './' indicates the element-wise division and $^\top$ denotes the transpose. Inequalities and the 'max' operation for a vector are applied element-wise.

## 2.3 Model Selection by Likelihood Cross Validation

The performance of KLIEP depends on the choice of basis functions $\{\varphi_\ell(\boldsymbol{x})\}_{\ell=1}^{b}$. Here we explain how they can be appropriately chosen from data samples.

Since KLIEP is based on the maximization of the score $J$ (see Eq.(2)), it would be natural to select the model such that $J$ is maximized. The expectation over $p_{\mathrm{te}}(\boldsymbol{x})$ involved in $J$ can be numerically approximated by *likelihood cross validation* (LCV) as follows: First, divide the test samples $\{\boldsymbol{x}_j^{\mathrm{te}}\}_{j=1}^{n_{\mathrm{te}}}$ into $R$ disjoint subsets $\{\mathcal{X}_r^{\mathrm{te}}\}_{r=1}^{R}$. Then obtain an importance estimate $\widehat{w}_r(\boldsymbol{x})$ from $\{\mathcal{X}_j^{\mathrm{te}}\}_{j \neq r}$ and approximate the score $J$ using $\mathcal{X}_r^{\mathrm{te}}$ as

$$\widehat{J}_r = \frac{1}{|\mathcal{X}_r^{\mathrm{te}}|}\sum_{\boldsymbol{x} \in \mathcal{X}_r^{\mathrm{te}}} \log \widehat{w}_r(\boldsymbol{x}).$$

We repeat this procedure for $r = 1, 2, \ldots, R$, compute the average of $\widehat{J}_r$ over all $r$, and use the average $\widehat{J}$ as an estimate of $J$:

$$\widehat{J} = \frac{1}{R}\sum_{r=1}^{R} \widehat{J}_r. \tag{4}$$

For model selection, we compute $\widehat{J}$ for all model candidates (the basis functions $\{\varphi_\ell(\boldsymbol{x})\}_{\ell=1}^{b}$ in the current setting) and choose the one that minimizes $\widehat{J}$. A pseudo code of the LCV procedure is summarized in Figure 1-(b)

One of the potential limitations of CV in general is that it is not reliable in small sample cases since data splitting by CV further reduces the sample size. On the other hand, in our CV procedure, the data splitting is performed over the *test input samples*, not over the training samples. Since we typically have a large number of test input samples, our CV procedure does not suffer from the small sample problem.

A good model may be chosen by the above CV procedure, given that a set of promising model candidates is prepared. As model candidates, we propose using a Gaussian kernel model centered at the *test* input points $\{\boldsymbol{x}_j^{\mathrm{te}}\}_{j=1}^{n_{\mathrm{te}}}$, i.e.,

$$\widehat{w}(\boldsymbol{x}) = \sum_{\ell=1}^{n_{\mathrm{te}}} \alpha_\ell K_\sigma(\boldsymbol{x}, \boldsymbol{x}_\ell^{\mathrm{te}}),$$

where $K_\sigma(\boldsymbol{x}, \boldsymbol{x}')$ is the Gaussian kernel with kernel width $\sigma$:

$$K_\sigma(\boldsymbol{x}, \boldsymbol{x}') = \exp\left\{-\frac{\|\boldsymbol{x} - \boldsymbol{x}'\|^2}{2\sigma^2}\right\}. \tag{5}$$

The reason why we chose the test input points $\{\boldsymbol{x}_j^{\text{te}}\}_{j=1}^{n_{\text{te}}}$ as the Gaussian centers, not the training input points $\{\boldsymbol{x}_i^{\text{tr}}\}_{i=1}^{n_{\text{tr}}}$, is as follows. By definition, the importance $w(\boldsymbol{x})$ tends to take large values if the training input density $p_{\text{tr}}(\boldsymbol{x})$ is small and the test input density $p_{\text{te}}(\boldsymbol{x})$ is large; conversely, $w(\boldsymbol{x})$ tends to be small (i.e., close to zero) if $p_{\text{tr}}(\boldsymbol{x})$ is large and $p_{\text{te}}(\boldsymbol{x})$ is small. When a function is approximated by a Gaussian kernel model, many kernels may be needed in the region where the output of the target function is large; on the other hand, only a small number of kernels would be enough in the region where the output of the target function is close to zero. Following this heuristic, we decided to allocate many kernels at high *test* input density regions, which can be achieved by setting the Gaussian centers at the test input points $\{\boldsymbol{x}_j^{\text{te}}\}_{j=1}^{n_{\text{te}}}$.

Alternatively, we may locate $(n_{\text{tr}} + n_{\text{te}})$ Gaussian kernels at both $\{\boldsymbol{x}_i^{\text{tr}}\}_{i=1}^{n_{\text{tr}}}$ and $\{\boldsymbol{x}_j^{\text{te}}\}_{j=1}^{n_{\text{te}}}$. However, in our preliminary experiments, this did not further improve the performance, but slightly increased the computational cost. Since $n_{\text{te}}$ is typically very large, just using all the test input points $\{\boldsymbol{x}_j^{\text{te}}\}_{j=1}^{n_{\text{te}}}$ as Gaussian centers is already computationally rather demanding. To ease this problem, we practically propose using a subset of $\{\boldsymbol{x}_j^{\text{te}}\}_{j=1}^{n_{\text{te}}}$ as Gaussian centers for computational efficiency, i.e.,

$$\widehat{w}(\boldsymbol{x}) = \sum_{\ell=1}^{b} \alpha_\ell K_\sigma(\boldsymbol{x}, \boldsymbol{c}_\ell), \tag{6}$$

where $\boldsymbol{c}_\ell$ is a template point randomly chosen from $\{\boldsymbol{x}_j^{\text{te}}\}_{j=1}^{n_{\text{te}}}$ and $b \ (\leq n_{\text{te}})$ is a prefixed number. In the rest of this paper, we fix the number of template points at

$$b = \min(100, n_{\text{te}}),$$

and optimize the kernel width $\sigma$ by the above CV procedure.

## 3 Experiments

In this section, we compare the experimental performance of KLIEP and existing approaches.

### 3.1 Importance Estimation for Artificial Data Sets

Let $p_{\text{tr}}(\boldsymbol{x})$ be the $d$-dimensional Gaussian density with mean $(0, 0, \ldots, 0)^\top$ and covariance identity and $p_{\text{te}}(\boldsymbol{x})$ be the $d$-dimensional Gaussian density with mean $(1, 0, \ldots, 0)^\top$ and covariance identity. The task is to estimate the importance at training input points:

$$w_i = w(\boldsymbol{x}_i^{\text{tr}}) = \frac{p_{\text{te}}(\boldsymbol{x}_i^{\text{tr}})}{p_{\text{tr}}(\boldsymbol{x}_i^{\text{tr}})} \quad \text{for } i = 1, 2, \ldots, n_{\text{tr}}.$$

We compare the following methods:

**KLIEP($\sigma$):** $\{w_i\}_{i=1}^{n_{\text{tr}}}$ are estimated by KLIEP with the Gaussian kernel model (6). Since the performance of KLIEP is dependent on the kernel width $\sigma$, we test several different values of $\sigma$.

**KLIEP(CV):** The kernel width $\sigma$ in KLIEP is chosen based on 5-fold LCV (see Section 2.3).

**KDE(CV):** $\{w_i\}_{i=1}^{n_{\text{tr}}}$ are estimated through the kernel density estimator (KDE) with the Gaussian kernel. The kernel widths for the training and test densities are chosen separately based on 5-fold likelihood cross-validation.

**KMM($\sigma$):** $\{w_i\}_{i=1}^{n_{\text{tr}}}$ are estimated by kernel mean matching (KMM) [6]. The performance of KMM is dependent on tuning parameters such as $B$, $\epsilon$, and $\sigma$. We set $B = 1000$ and $\epsilon = (\sqrt{n_{\text{tr}}} - 1)/\sqrt{n_{\text{tr}}}$ following the paper [6], and test several different values of $\sigma$. We used the *CPLEX* software for solving quadratic programs in the experiments.

**LogReg($\sigma$):** Importance weights are estimated by logistic regression (LogReg) [2]. The Gaussian kernels are used as basis functions. Since the performance of LogReg is dependent on the kernel width $\sigma$, we test several different values of $\sigma$. We used the *LIBLINEAR* implementation of logistic regression for the experiments [7].

**LogReg(CV):** The kernel width $\sigma$ in LogReg is chosen based on 5-fold CV.

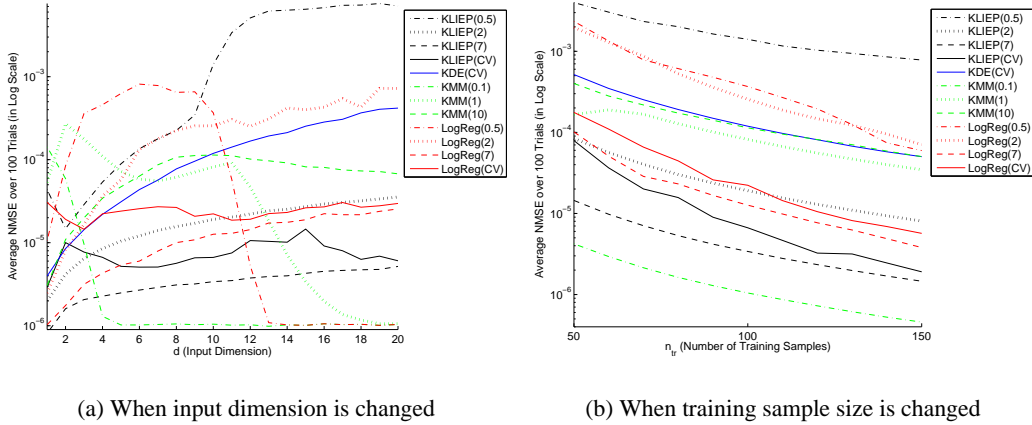

(a) When input dimension is changed  (b) When training sample size is changed

Figure 2: NMSEs averaged over 100 trials in log scale.

We fixed the number of test input points at $n_{\text{te}} = 1000$ and consider the following two settings for the number $n_{\text{tr}}$ of training samples and the input dimension $d$:

**(a)** $n_{\text{tr}} = 100$ and $d = 1, 2, \ldots, 20$,

**(b)** $d = 10$ and $n_{\text{tr}} = 50, 60, \ldots, 150$.

We run the experiments 100 times for each $d$, each $n_{\text{tr}}$, and each method, and evaluate the quality of the importance estimates $\{\widehat{w}_i\}_{i=1}^{n_{\text{tr}}}$ by the *normalized mean squared error* (NMSE):

$$\text{NMSE} = \frac{1}{n_{\text{tr}}} \sum_{i=1}^{n_{\text{tr}}} \left( \frac{\widehat{w}_i}{\sum_{i'=1}^{n_{\text{tr}}} \widehat{w}_{i'}} - \frac{w_i}{\sum_{i'=1}^{n_{\text{tr}}} w_{i'}} \right)^2.$$

NMSEs averaged over 100 trials are plotted in log scale in Figure 2. Figure 2(a) shows that the error of KDE(CV) sharply increases as the input dimension grows, while KLIEP, KMM, and LogReg with appropriate kernel widths tend to give smaller errors than KDE(CV). This would be the fruit of directly estimating the importance without going through density estimation. The graph also show that the performance of KLIEP, KMM, and LogReg is dependent on the kernel width $\sigma$—the results of KLIEP(CV) and LogReg(CV) show that model selection is carried out reasonably well and KLIEP(CV) works significantly better than LogReg(CV).

Figure 2(b) shows that the errors of all methods tend to decrease as the number of training samples grows. Again, KLIEP, KMM, and LogReg with appropriate kernel widths tend to give smaller errors than KDE(CV). Model selection in KLIEP(CV) and LogReg(CV) works reasonably well and KLIEP(CV) tends to give significantly smaller errors than LogReg(CV).

Overall, KLIEP(CV) is shown to be a useful method in importance estimation.

### 3.2 Covariate Shift Adaptation with Regression and Classification Benchmark Data Sets

Here we employ importance estimation methods for covariate shift adaptation in regression and classification benchmark problems (see Table 1).

Each data set consists of input/output samples $\{(\boldsymbol{x}_k, y_k)\}_{k=1}^n$. We normalize all the input samples $\{\boldsymbol{x}_k\}_{k=1}^n$ into $[0,1]^d$ and choose the test samples $\{(\boldsymbol{x}_j^{\text{te}}, y_j^{\text{te}})\}_{j=1}^{n_{\text{te}}}$ from the pool $\{(\boldsymbol{x}_k, y_k)\}_{k=1}^n$ as follows. We randomly choose one sample $(\boldsymbol{x}_k, y_k)$ from the pool and accept this with probability $\min(1, 4(x_k^{(c)})^2)$, where $x_k^{(c)}$ is the $c$-th element of $\boldsymbol{x}_k$ and $c$ is randomly determined and fixed in each trial of experiments; then we remove $\boldsymbol{x}_k$ from the pool regardless of its rejection or acceptance, and repeat this procedure until we accept $n_{\text{te}}$ samples. We choose the training samples $\{(\boldsymbol{x}_i^{\text{tr}}, y_i^{\text{tr}})\}_{i=1}^{n_{\text{tr}}}$ uniformly from the rest. Intuitively, in this experiment, the test input density tends

to be lower than the training input density when $x_k^{(c)}$ is small. We set the number of samples at $n_{\mathrm{tr}} = 100$ and $n_{\mathrm{te}} = 500$ for all data sets. Note that we only use $\{(\boldsymbol{x}_i^{\mathrm{tr}}, y_i^{\mathrm{tr}})\}_{i=1}^{n_{\mathrm{tr}}}$ and $\{\boldsymbol{x}_j^{\mathrm{te}}\}_{j=1}^{n_{\mathrm{te}}}$ for training regressors or classifiers; the test output values $\{y_j^{\mathrm{te}}\}_{j=1}^{n_{\mathrm{te}}}$ are used only for evaluating the generalization performance.

We use the following kernel model for regression or classification:

$$\widehat{f}(\boldsymbol{x}; \boldsymbol{\theta}) = \sum_{\ell=1}^{t} \theta_\ell K_h(\boldsymbol{x}, \boldsymbol{m}_\ell),$$

where $K_h(\boldsymbol{x}, \boldsymbol{x}')$ is the Gaussian kernel (5) and $\boldsymbol{m}_\ell$ is a template point randomly chosen from $\{\boldsymbol{x}_j^{\mathrm{te}}\}_{j=1}^{n_{\mathrm{te}}}$. We set the number of kernels at $t = 50$. We learn the parameter $\boldsymbol{\theta}$ by *importance-weighted regularized least squares* (IWRLS) [9]:

$$\widehat{\boldsymbol{\theta}}_{IWRLS} \equiv \operatorname*{argmin}_{\boldsymbol{\theta}} \left[ \sum_{i=1}^{n_{\mathrm{tr}}} \widehat{w}(\boldsymbol{x}_i^{\mathrm{tr}}) \left( \widehat{f}(\boldsymbol{x}_i^{\mathrm{tr}}; \boldsymbol{\theta}) - y_i^{\mathrm{tr}} \right)^2 + \lambda \|\boldsymbol{\theta}\|^2 \right]. \tag{7}$$

The solution $\widehat{\boldsymbol{\theta}}_{IWRLS}$ is analytically given by

$$\widehat{\boldsymbol{\theta}} = (\boldsymbol{K}^\top \widehat{\boldsymbol{W}} \boldsymbol{K} + \lambda \boldsymbol{I})^{-1} \boldsymbol{K}^\top \widehat{\boldsymbol{W}} \boldsymbol{y},$$

where $\boldsymbol{I}$ is the identity matrix and

$$\boldsymbol{y} = (y_1, y_2, \ldots, y_{n_{\mathrm{tr}}})^\top,$$
$$K_{i,\ell} = K_h(\boldsymbol{x}_i^{\mathrm{tr}}, \boldsymbol{m}_\ell),$$
$$\widehat{\boldsymbol{W}} = \operatorname{diag}\left(\widehat{w}_1, \widehat{w}_2, \ldots, \widehat{w}_{n_{\mathrm{tr}}}\right).$$

The kernel width $h$ and the regularization parameter $\lambda$ in IWRLS (7) are chosen by 5-fold importance weighted CV (IWCV) [9]. We compute the IWCV score by

$$\frac{1}{|\mathcal{Z}_r^{\mathrm{tr}}|} \sum_{(\boldsymbol{x}, y) \in \mathcal{Z}_r^{\mathrm{tr}}} \widehat{w}(\boldsymbol{x}) L\left(\widehat{f}_r(\boldsymbol{x}), y\right),$$

where

$$L\left(\widehat{y}, y\right) = \begin{cases} (\widehat{y} - y)^2 & \text{(Regression)}, \\ \frac{1}{2}(1 - \operatorname{sign}\{\widehat{y}y\}) & \text{(Classification)}. \end{cases}$$

We run the experiments 100 times for each data set and evaluate the *mean test error*:

$$\frac{1}{n_{\mathrm{te}}} \sum_{j=1}^{n_{\mathrm{te}}} L\left(\widehat{f}(\boldsymbol{x}_j^{\mathrm{te}}), y_j^{\mathrm{te}}\right).$$

The results are summarized in Table 1, where 'Uniform' denotes uniform weights, i.e., no importance weight is used. The table shows that KLIEP(CV) compares favorably with Uniform, implying that the importance weighted methods combined with KLIEP(CV) are useful for improving the prediction performance under covariate shift. KLIEP(CV) works much better than KDE(CV); actually KDE(CV) tends to be worse than Uniform, which may be due to high dimensionality. We tested 10 different values of the kernel width $\sigma$ for KMM and described three representative results in the table. KLIEP(CV) is slightly better than KMM with the best kernel width. Finally, LogReg(CV) works reasonably well, but it sometimes performs poorly.

Overall, we conclude that the proposed KLIEP(CV) is a promising method for covariate shift adaptation.

## 4 Conclusions

In this paper, we addressed the problem of estimating the importance for covariate shift adaptation. The proposed method, called KLIEP, does not involve density estimation so it is more advantageous than a naive KDE-based approach particularly in high-dimensional problems. Compared with KMM

Table 1: Mean test error averaged over 100 trials. The numbers in the brackets are the standard deviation. All the error values are normalized so that the mean error by 'Uniform' (uniform weighting, or equivalently no importance weighting) is one. For each data set, the best method and comparable ones based on the *Wilcoxon signed rank test* at the significance level $5\%$ are described in bold face. The upper half are regression data sets taken from DELVE and the lower half are classification data sets taken from IDA. 'KMM($\sigma$)' denotes KMM with kernel width $\sigma$.

| Data | Dim | Uniform | KLIEP (CV) | KDE (CV) | KMM (0.01) | KMM (0.3) | KMM (1) | LogReg (CV) |
|------|-----|---------|-----------|----------|-----------|----------|---------|-------------|
| kin-8fh | 8 | 1.00(0.34) | **0.95(0.31)** | 1.22(0.52) | 1.00(0.34) | 1.12(0.37) | 1.59(0.53) | 1.30(0.40) |
| kin-8fm | 8 | 1.00(0.39) | **0.86(0.35)** | 1.12(0.57) | 1.00(0.39) | **0.98(0.46)** | 1.95(1.24) | 1.29(0.58) |
| kin-8nh | 8 | **1.00(0.26)** | 0.99(0.22) | 1.09(0.20) | **1.00(0.27)** | 1.04(0.17) | 1.16(0.25) | 1.06(0.17) |
| kin-8nm | 8 | **1.00(0.30)** | 0.97(0.25) | 1.14(0.26) | 1.00(0.30) | 1.09(0.23) | 1.20(0.22) | 1.13(0.25) |
| abalone | 7 | **1.00(0.50)** | 0.94(0.67) | 1.02(0.41) | 1.01(0.51) | **0.96(0.70)** | **0.93(0.39)** | **0.92(0.41)** |
| image | 18 | **1.00(0.51)** | 0.94(0.44) | 0.98(0.45) | **0.97(0.50)** | **0.97(0.45)** | 1.09(0.54) | **0.99(0.48)** |
| ringnorm | 20 | 1.00(0.04) | 0.99(0.06) | 0.87(0.04) | 1.00(0.04) | **0.87(0.05)** | **0.87(0.05)** | 0.95(0.08) |
| twonorm | 20 | 1.00(0.58) | **0.91(0.52)** | 1.16(0.71) | 0.99(0.50) | **0.86(0.55)** | **0.99(0.70)** | **0.94(0.59)** |
| waveform | 21 | **1.00(0.45)** | **0.93(0.34)** | 1.05(0.47) | **1.00(0.44)** | **0.93(0.32)** | 0.98(0.31) | **0.95(0.34)** |
| Average | | 1.00(0.38) | 0.94(0.35) | 1.07(0.40) | 1.00(0.36) | 0.98(0.37) | 1.20(0.47) | 1.06(0.37) |

which also directly gives importance estimates, KLIEP is practically more useful since it is equipped with a model selection procedure. Our experiments highlighted these advantages and therefore KLIEP is shown to be a promising method for covariate shift adaptation.

In KLIEP, we modeled the importance function by a linear (or kernel) model, which resulted in a convex optimization problem with a sparse solution. However, our framework allows the use of any models. An interesting future direction to pursue would be to search for a class of models which has additional advantages.

Finally, the range of application of importance weights is not limited to covariate shift adaptation. For example, the density ratio could be used for novelty detection. Exploring possible application areas will be important future directions.

### Acknowledgments

This work was supported by MEXT (17700142 and 18300057), the Okawa Foundation, the Microsoft CORE3 Project, and the IBM Faculty Award.

## References

[1] P. Baldi and S. Brunak. *Bioinformatics: The Machine Learning Approach*. MIT Press, Cambridge, 1998.

[2] S. Bickel, M. Brückner, and T. Scheffer. Discriminative learning for differing training and test distributions. In *Proceedings of the 24th International Conference on Machine Learning*, 2007.

[3] S. Bickel and T. Scheffer. Dirichlet-enhanced spam filtering based on biased samples. In B. Schölkopf, J. Platt, and T. Hoffman, editors, *Advances in Neural Information Processing Systems 19*. MIT Press, Cambridge, MA, 2007.

[4] S. Boyd and L. Vandenberghe. *Convex Optimization*. Cambridge University Press, Cambridge, 2004.

[5] J. J. Heckman. Sample selection bias as a specification error. *Econometrica*, 47(1):153–162, 1979.

[6] J. Huang, A. Smola, A. Gretton, K. M. Borgwardt, and B. Schölkopf. Correcting sample selection bias by unlabeled data. In B. Schölkopf, J. Platt, and T. Hoffman, editors, *Advances in Neural Information Processing Systems 19*, pages 601–608. MIT Press, Cambridge, MA, 2007.

[7] C.-J. Lin, R. C. Weng, and S. S. Keerthi. Trust region Newton method for large-scale logistic regression. Technical report, Department of Computer Science, National Taiwan University, 2007.

[8] H. Shimodaira. Improving predictive inference under covariate shift by weighting the log-likelihood function. *Journal of Statistical Planning and Inference*, 90(2):227–244, 2000.

[9] M. Sugiyama, M. Krauledat, and K.-R. Müller. Covariate shift adaptation by importance weighted cross validation. *Journal of Machine Learning Research*, 8:985–1005, May 2007.

[10] R. S. Sutton and G. A. Barto. *Reinforcement Learning: An Introduction*. MIT Press, Cambridge, MA, 1998.

